# Off-Road Obstacle Avoidance through End-to-End Learning

**Yann LeCun**
Courant Institute of Mathematical Sciences
New York University,
New York, NY 10004, USA
http://yann.lecun.com

**Urs Muller**
Net-Scale Technologies
Morganville, NJ 07751, USA
urs@net-scale.com

**Jan Ben**
Net-Scale Technologies
Morganville, NJ 07751, USA

**Eric Cosatto**
NEC Laboratories,
Princeton, NJ 08540

**Beat Flepp**
Net-Scale Technologies
Morganville, NJ 07751, USA

## Abstract

We describe a vision-based obstacle avoidance system for off-road mobile robots. The system is trained from end to end to map raw input images to steering angles. It is trained in supervised mode to predict the steering angles provided by a human driver during training runs collected in a wide variety of terrains, weather conditions, lighting conditions, and obstacle types. The robot is a 50cm off-road truck, with two forward-pointing wireless color cameras. A remote computer processes the video and controls the robot via radio. The learning system is a large 6-layer convolutional network whose input is a single left/right pair of unprocessed low-resolution images. The robot exhibits an excellent ability to detect obstacles and navigate around them in real time at speeds of 2 m/s.

## 1   Introduction

Autonomous off-road vehicles have vast potential applications in a wide spectrum of domains such as exploration, search and rescue, transport of supplies, environmental management, and reconnaissance. Building a fully autonomous off-road vehicle that can reliably navigate and avoid obstacles at high speed is a major challenge for robotics, and a new domain of application for machine learning research.

The last few years have seen considerable progress toward that goal, particularly in areas such as mapping the environment from active range sensors and stereo cameras [11, 7], simultaneously navigating and building maps [6, 15], and classifying obstacle types.

Among the various sub-problems of off-road vehicle navigation, obstacle detection and avoidance is a subject of prime importance. The wide diversity of appearance of potential obstacles, and the variability of the surroundings, lighting conditions, and other factors, make the problem very challenging.

Many recent efforts have attacked the problem by relying on a multiplicity of sensors, including laser range finder and radar [11]. While active sensors make the problem considerably simpler, there seems to be an interest from potential users for purely passive systems that rely exclusively on camera input. Cameras are considerably less expensive,

bulky, power hungry, and detectable than active sensors, allowing levels of miniaturization that are not otherwise possible. More importantly, active sensors can be slow, limited in range, and easily confused by vegetation, despite rapid progress in the area [2].

Avoiding obstacles by relying solely on camera input requires solving a highly complex vision problem. A time-honored approach is to derive range maps from multiple images through multiple cameras or through motion [6, 5]. Deriving steering angles to avoid obstacles from the range maps is a simple matter. A large number of techniques have been proposed in the literature to construct range maps from stereo images. Such methods have been used successfully for many years for navigation in indoor environments where edge features can be reliably detected and matched [1], but navigation in outdoors environment, despite a long history, is still a challenge [14, 3]: real-time stereo algorithms are considerably less reliable in unconstrained outdoors environments. The extreme variability of lighting conditions, and the highly unstructured nature of natural objects such as tall grass, bushes and other vegetation, water surfaces, and objects with repeating textures, conspire to limit the reliability of this approach. In addition, stereo-based methods have a rather limited range, which dramatically limits the maximum driving speed.

## 2    End-To-End Learning for Obstacle Avoidance

In general, computing depth from stereo images is an ill-posed problem, but the depth map is only a means to an end. Ultimately, the output of an obstacle avoidance system is a set of possible steering angles that direct the robot toward traversible regions.

Our approach is to view the entire problem of mapping input stereo images to possible steering angles as a single indivisible task to be learned *from end to end*. Our learning system takes raw color images from two forward-pointing cameras mounted on the robot, and maps them to a set of possible steering angles through a single trained function.

The training data was collected by recording the actions of a human driver together with the video data. The human driver remotely drives the robot straight ahead until the robot encounters a non-traversible obstacle. The human driver then avoids the obstacle by steering the robot in the appropriate direction. The learning system is trained in supervised mode. It takes a single pair of heavily-subsampled images from the two cameras, and is trained to predict the steering angle produced by the human driver at that time.

The learning architecture is a 6-layer convolutional network [9]. The network takes the left and right $149 \times 58$ color images and produces two outputs. A large value on the first output is interpreted as a left steering command while a large value on the second output indicates a right steering command. Each layer in a convolutional network can be viewed as a set of trainable, shift-invariant linear filters with local support, followed by a point-wise non-linear saturation function. All the parameters of all the filters in the various layers are trained simultaneously. The learning algorithm minimizes the discrepancy between the desired output vector and the output vector produced by the output layer.

The approach is somewhat reminiscent of the ALVINN and MANIAC systems [13, 4]. The main differences with ALVINN are: (1) our system uses stereo cameras; (2) it is trained for off-road obtacle avoidance rather than road following; (3) Our trainable system uses a convolutional network rather than a traditional fully-connected neural net.

Convolutional networks have two considerable advantages for this applications. Their local and sparse connection scheme allows us to handle images of higher resolution than ALVINN while keeping the size of the network within reasonnable limits. Convolutional nets are particularly well suited for our task because local feature detectors that combine inputs from the left and right images can be useful for estimating distances to obstacles (possibly by estimating disparities). Furthermore, the local and shift-invariant property of the filters allows the system to learn relevant local features with a limited amount of training data.

They key advantage of the approach is that the entire function from raw pixels to steering angles is trained from data, which completely eliminates the need for feature design and

selection, geometry, camera calibration, and hand-tuning of parameters. The main motivation for the use of end-to-end learning is, in fact, to eliminate the need for hand-crafted heuristics. Relying on automatic global optimization of an objective function from massive amounts for data may produce systems that are more robust to the unpredictable variability of the real world. Another potential benefit of a pure learning-based approach is that the system may use other cues than stereo disparity to detect obstacles, possibly alleviating the short-sightedness of methods based purely on stereo matching.

## 3   Vehicle Hardware

We built a small and light-weight vehicle which can be carried by a single person so as to facilitate data collection and testing in a wide variety of environments. Using a small, rugged and low-cost robot allowed us to drive at relatively high speed without fear of causing damage to people, property or the robot itself. The downside of this approach is the limited payload, too limited for holding the computing power necessary for the visual processing. Therefore, the robot has no significant on-board computing power. It is remotely controled by an off-board computer. A wireless link is used to transmit video and sensor readings to the remote computer. Throttle and steering controls are sent from the computer to the robot through a regular radio control channel.

The robot chassis was built around a customized 1/10-th scale remote-controlled, electric-powered, four-wheel-drive truck which was roughly 50cm in length. The typical speed of the robot during data collection and testing sessions was roughly 2 meters per second. Two forward-pointing low-cost 1/3-inch CCD cameras were mounted 110mm apart behind a clear lexan window. With 2.5mm lenses, the horizontal field of view of each camera was about 100 degrees.

A pair of 900MHz analog video transmitters was used to send the camera outputs to the remote computer. The analog video links were subject to high signal noise, color shifts, frequent interferences, and occasional video drop-outs. But the small size, light weight, and low cost provided clear advantages. The vehicle is shown in Figure 1. The remote control station consisted of a 1.4GHz Athlon PC running Linux with video capture cards, and an interface to an R/C transmitter.

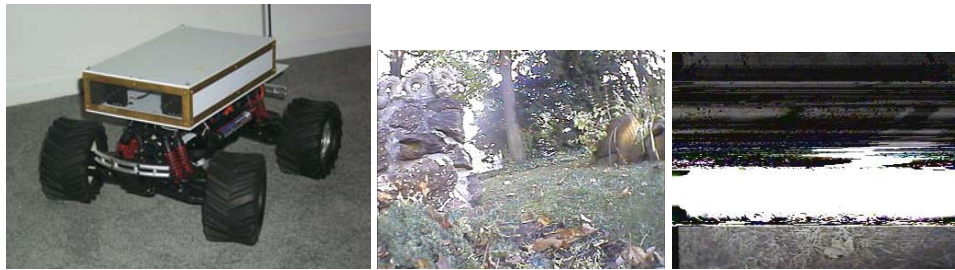

Figure 1: Left: The robot is a modified 50 cm-long truck platform controled by a remote computer. Middle: sample images images from the training data. Right: poor reception occasionally caused bad quality images.

## 4   Data Collection

During a data collection session, the human operator wears video goggles fed with the video signal from one the robot's cameras (no stereo), and controls the robot through a joystick connected to the PC. During each run, the PC records the output of the two video cameras at 15 frames per second, together with the steering angle and throttle setting from the operator.

A crucially important requirement of the data collection process was to collect large amounts of data with enough diversity of terrain, obstacles, and lighting conditions. Tt was necessary for the human driver to adopt a *consistent* obstacle avoidance behaviour. To ensure this, the human driver was to drive the vehicle straight ahead whenever no obstacle was present within a threatening distance. Whenever the robot approached an obstacle, the human driver had to steer left or right so as to avoid the obstacle. The general strategy for collecting training data was as follows: (a) Collecting data from as large a variety of off-road training grounds as possible. Data was collected from a large number of parks, playgrounds, frontyards and backyards of a number of suburban homes, and heavily cluttered construction areas; (b) Collecting data with various lighting conditions, i. e., different weather conditions and different times of day; (c) Collecting sequences where the vehicle starts driving straight and then is steered left or right as the robot approached an obstacle; (d) Avoiding turns when no obstacles were present; (e) Including straight runs with no obstacles and no turns as part of the training set; (f) Trying to be consistent in the turning behavior, i. e., always turning at approximately the same distance from an obstacle.

Even though great care was taken in collecting the highest quality training data, there were a number of imperfections in the training data that could not be avoided: (a) The small-form-factor, low-cost cameras presented significant differences in their default settings. In particular, the white balance of the two cameras were somewhat different; (b) To maximize image quality, the automatic gain control and automatic exposure were activated. Because of differences in fabrication, the left and right images had slightly different brightness and contrast characteristics. In particular, the AGC adjustments seem to react at different speeds and amplitudes; (c) Because of AGC, driving into the sunlight caused the images to become very dark and obstacles to become hard to detect; (d) The wireless video connection caused dropouts and distortions of some frames. Approximately 5 % of the frames were affected. An example is shown in Figures 1; (e) The cameras were mounted rigidly on the vehicle and were exposed to vibration, despite the suspension. Despite these difficult conditions, the system managed to learn the task quite well as will be shown later.

The data was recorded and archived at a resolution of $320 \times 240 \times$ pixels at 15 frames per second. The data was collected on 17 different days during the Winter of 2003/2004 (the sun was very low on the horizon). A total of 1,500 clips were collected with an average length of about 85 frames each. This resulted in a total of about 127,000 individual pairs of frames. Segments during which the robot was driven into position in preparation for a run were edited out. No other manual data cleaning took place. In the end, 95,000 frame pairs were used for training and 32,000 for validation/testing. The training pairs and testing pairs came from different sequences (and often different locations).

Figure 1 shows example snapshots from the training data, including an image with poor reception. Note that only one of the two (stereo) images is shown. High noise and frame dropouts occurred in approximately 5 % of the frames. It was decided to leave them in the training set and test set so as to train the system under realistic conditions.

## 5 The Learning System

The entire processing consists of a single convolutional network. The architecture of convolutional nets is somewhat inspired by the structure of biological visual systems. Convolutional nets have been used successfully in a number of vision applications such as handwriting recognition [9], object recognition [10], and face detection [12].

The input to the convolutional net consists of 6 planes of size $149 \times 58$ pixels. The six planes respectively contain the Y, U and V components for the left camera and the right camera. The input images were obtained by cropping the $320 \times 240$ images, and through $2\times$ horizontal low-pass filtering and subsampling, and $4\times$ vertical low-pass filtering and subsampling. The horizontal resolution was set higher so as to preserve more accurate image disparity information.

Each layer in a convolutional net is composed of units organized in planes called feature maps. Each unit in a feature map takes inputs from a small neighborhood within the feature

maps of the previous layer. Neighborhing units in a feature map are connected to neighboring (possibly overlapping) windows. Each unit computes a weighted sum of its inputs and passes the result through a sigmoid saturation function. All units within a feature map share the same weights. Therefore, each feature map can be seen as convolving the feature maps of the previous layers with small-size kernels, and passing the sum of those convolutions through sigmoid functions. Units in a feature map detect local features at all locations on the previous layer.

The first layer contains 6 feature maps of size $147{\times}56$ connected to various combinations of the input maps through $3{\times}3$ kernels. The first feature map is connected to the YUV planes of the left image, the second feature map to the YUV planes of the right image, and the other 4 feature maps to all 6 input planes. Those 4 feature maps are binocular, and can learn filters that compare the location of features in the left and right images. Because of the weight sharing, the first layer merely has 276 free parameters (30 kernels of size $3{\times}3$ plus 6 biases). The next layer is an averaging/subsampling layer of size $49{\times}14$ whose purpose is to reduce the spatial resolution of the feature maps so as to build invariances to small geometric distortions of the input. The subsampling ratios are 3 horizontally and 4 vertically. The 3-rd layer contains 24 feature maps of size $45{\times}12$. Each feature map is connected to various subsests of maps in the previous layer through a total of 96 kernels of size $5{\times}3$. The 4-th layer is an averaging/subsampling layer of size $9{\times}4$ with $5{\times}3$ subsampling ratios. The 5-th layer contains 100 feature maps of size $1{\times}1$ connected to the 4-th layer through 2400 kernels of size $9{\times}4$ (full connection). finally, the output layer contains two units fully-connected to the 100 units in the 5-th layer. The two outputs respectively code for "turn left" and "turn right" commands. The network has 3.15 Million connections and about 72,000 trainable parameters.

The bottom half of figure 2 shows the states of the six layers of the convolutional net. the size of the input, $149{\times}58$, was essentially limited by the computing power of the remote computer (a 1.4GHz Athlon). The network as shown runs in about 60ms per image pair on the remote computer. Including all the processing, the driving system ran at a rate of 10 cycles per second.

The system's output is computed on a frame by frame basis with no memory of the past and no time window. Using multiple successive frames as input would seem like a good idea since the multiple views resulting from ego-motion facilitates the segmentation and detection of nearby obstacles. Unfortunately, the supervised learning approach precludes the use of multiple frames. The reason is that since the steering is fairly smooth in time (with long, stable periods), the current rate of turn is an excellent predictor of the next desired steering angle. But the current rate of turn is easily derived from multiple successive frames. Hence, a system trained with multiple frames would merely predict a steering angle equal to the current rate of turn as observed through the camera. This would lead to catastrophic behavior in test mode. The robot would simply turn in circles.

The system was trained with a stochastic gradient-based method that automatically sets the relative step sizes of the parameters based on the local curvature of the loss surface [8]. Gradients were computed using the variant of back-propagation appropriate for convolutional nets.

# 6 Results

Two performance measurements were recorded, the average loss, and the percentage of "correctly classified" steering angles. The average loss is the sum of squared differences between outputs produced by the system and the target outputs, averaged over all samples. The percentage of correctly classified steering angles measures the number of times the predicted steering angle, quantized into three bins (left, straight, right), agrees with steering angle provided by the human driver. Since the thresholds for deciding whether an angle counted as left, center, or right were somewhat arbitrary, the percentages cannot be intepreted in absolute terms, but merely as a relative figure of merit for comparing runs and architectures.

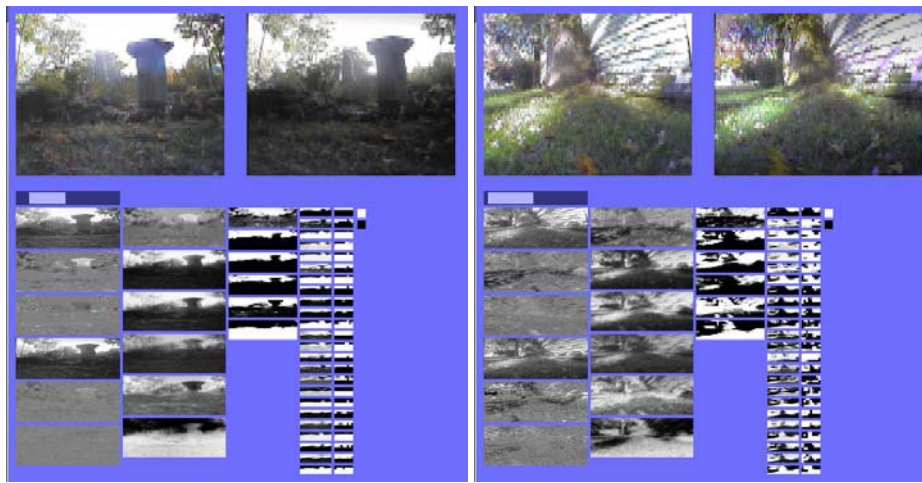

Figure 2: Internal state of the convolutional net for two sample frames. The top row shows left/right image pairs extracted from the test set. The light-blue bars below show the steering angle produced by the system. The bottom halves show the state of the layers of the network, where each column is a layer (the penultimate layer is not shown). Each rectangular image is a feature map in which each pixel represents a unit activation. The YUV components of the left and right input images are in the leftmost column.

With 95,000 training image pairs, training took 18 epochs through the training set. No significant improvements in the error rate occurred thereafter. After training, the error rate was 25.1% on the training set, and 35.8% on the test set. The average loss (mean-sqaured error) was 0.88 on the training set and 1.24 on the test set. A complete training session required about four days of CPU time on a 3.0GHz Pentium/Xeon-based server. Naturally, a classification error rate of 35.8 % doesn't mean that the vehicle crashes into obstacles 35.8 % of the time, but merely that the prediction of the system was in a different bin than that of the human drivers for 35.8 % of the frames. The seemingly high error rate is not an accurate reflection of the actual effectiveness of the robot in the field. There are several reasons for this. First, there may be several legitimate steering angles for a given image pair: turning left or right around an obstacle may both be valid options, but our performance measure would record one of those options as incorrect. In addition, many illegitimate errors are recorded when the system starts turning at a different time than the human driver, or when the precise values of the steering angles are different enough to be in different bins, but close enough to cause the robot to avoid the obstacle. Perhaps more informative is diagram in figure 3. It shows the steering angle produced by the system and the steering angle provided by the human driver for 8000 frames from the test set. It is clear for the plot that only a small number of obstacles would not have been avoided by the robot.

The best performance measure is a set of actual runs through representative testing grounds. Videos of typical test runs are available at **http://www.cs.nyu.edu/~yann/research/dave/index.html**.

Figure 2 shows a snapshot of the trained system in action. The network was presented with a scene that was not present in the training set. This figure shows that the system can detect obstacles and predict appropriate steering angles in the presence of back-lighting and with wild difference between the automatics gain settings of the left and right cameras.

Another visualization of the results can be seen in Figures 4. They are snapshots of video clips recorded from the vehicle's cameras while the vehicle was driving itself autonomously. Only one of the two camera outputs is shown here. Each picture also shows

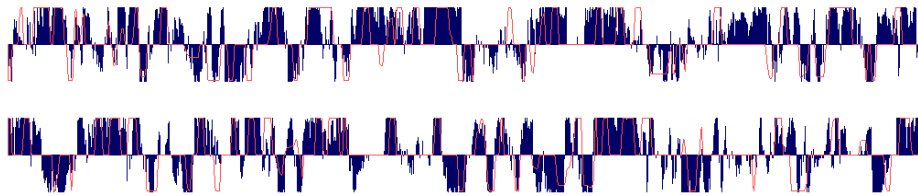

Figure 3: The steering angle produced by the system (black) compared to the steering angle provided by the human operator (red line) for 8000 frames from the test set. Very few obstacles would not have been avoided by the system.

the steering angle produced by the system for that particular input.

# 7    Conclusion

We have demonstrate the applicability of end-to-end learning methods to the task of obstacle avoidance for off-road robots.

A 6-layer convolutional network was trained with massive amounts of data to emulate the obstacle avoidance behavior of a human driver. the architecture of the system allowed it to learn low-level and high-level features that reliably predicted the bearing of traversible areas in the visual field.

The main advantage of the system is its robustness to the extreme diversity of situations in off-road environments. Its main design advantage is that it is trained from raw pixels to directly produce steering angles. The approach essentially eliminates the need for manual calibration, adjustments, parameter tuning etc. Furthermore, the method gets around the need to design and select an appropriate set of feature detectors, as well as the need to design robust and fast stereo algorithms.

The construction of a fully autonomous driving system for ground robots will require several other components besides the purely-reactive obstacle detection and avoidance system described here. The present work is merely one component of a future system that will include map building, visual odometry, spatial reasoning, path finding, and other strategies for the identification of traversable areas.

### Acknowledgment

This project was a preliminary study for the DARPA project "Learning Applied to Ground Robots" (LAGR). The material presented is based upon work supported by the Defense Advanced Research Project Agency Information Processing Technology Office, ARPA Order No. Q458, Program Code No. 3D10, Issued by DARPA/CMO under Contract #MDA972-03-C-0111.

# References

[1] N. Ayache and O. Faugeras. Maintaining representations of the environment of a mobile robot. *IEEE Trans. Robotics and Automation*, 5(6):804–819, 1989.

[2] C. Bergh, B. Kennedy, L. Matthies, and Johnson A. A compact, low power two-axis scanning laser rangefinder for mobile robots. In *The 7th Mechatronics Forum International Conference*, 2000.

[3] S. B. Goldberg, M. Maimone, and L. Matthies. Stereo vision and rover navigation software for planetary exploration. In *IEEE Aerospace Conference Proceedings*, March 2002.

[4] T. Jochem, D. Pomerleau, and C. Thorpe. Vision-based neural network road and intersection detection and traversal. In *Proc. IEEE Conf. Intelligent Robots and Systems*, volume 3, pages 344–349, August 1995.

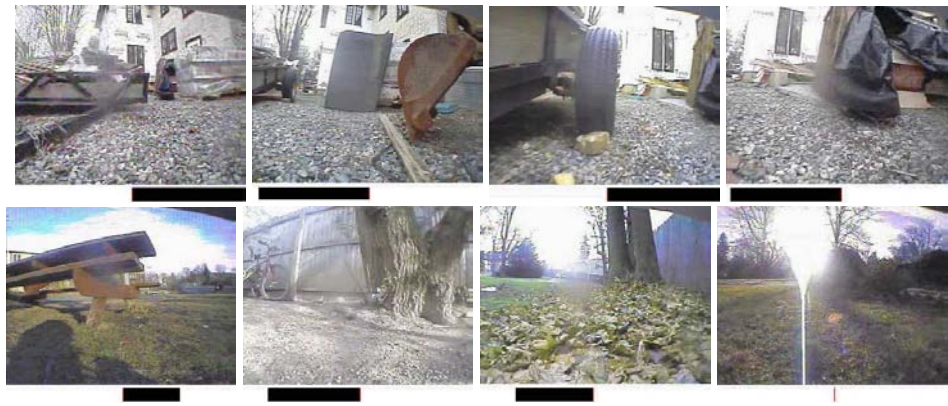

Figure 4: Snapshots from the left camera while the robots drives itself through various environment. The black bar beneath each image indicates the steering angle produced by the system. Top row: four successive snapshots showing the robot navigating through a narrow passageway between a trailer, a backhoe, and some construction material. Bottom row, left: narrow obstacles such as table legs and poles (left), and solid obstacles such as fences (center-left) are easily detected and avoided. Higly textured objects on the ground do not detract the system from the correct response (center-right). One scenario where the vehicle occasionally made wrong decisions is when the sun is in the field of view: the system seems to systematically drive towards the sun, whenever the sun is low on the horizon (right). Videos of these sequences are available at **http://www.cs.nyu.edu/˜yann/research/dave/index.html**.

[5] A. Kelly and A. Stentz. Stereo vision enhancements for low-cost outdoor autonomous vehicles. In *International Conference on Robotics and Automation, Workshop WS-7, Navigation of Outdoor Autonomous Vehicles, (ICRA '98)*, May 1998.

[6] D.J. Kriegman, E. Triendl, and T.O. Binford. Stereo vision and navigation in buildings for mobile robots. *IEEE Trans. Robotics and Automation*, 5(6):792–803, 1989.

[7] E. Krotkov and M. Hebert. Mapping and positioning for a prototype lunar rover. In *Proc. IEEE Int'l Conf. Robotics and Automation*, pages 2913–2919, May 1995.

[8] Y. LeCun, L. Bottou, G. Orr, and K. Muller. Efficient backprop. In G. Orr and Muller K., editors, *Neural Networks: Tricks of the trade*. Springer, 1998.

[9] Yann LeCun, Leon Bottou, Yoshua Bengio, and Patrick Haffner. Gradient-based learning applied to document recognition. *Proceedings of the IEEE*, 86(11):2278–2324, November 1998.

[10] Yann LeCun, Fu-Jie Huang, and Leon Bottou. Learning methods for generic object recognition with invariance to pose and lighting. In *Proceedings of CVPR'04*. IEEE Press, 2004.

[11] L. Matthies, E. Gat, R. Harrison, B. Wilcox, R. Volpe, and T. Litwin. Mars microrover navigation: Performance evaluation and enhancement. In *Proc. IEEE Int'l Conf. Intelligent Robots and Systems*, volume 1, pages 433–440, August 1995.

[12] R. Osadchy, M. Miller, and Y. LeCun. Synergistic face detection and pose estimation with energy-based model. In *Advances in Neural Information Processing Systems (NIPS 2004)*. MIT Press, 2005.

[13] Dean A. Pomerleau. Knowledge-based training of artificial neural netowrks for autonomous robot driving. In J. Connell and S. Mahadevan, editors, *Robot Learning*. Kluwer Academic Publishing, 1993.

[14] C. Thorpe, M. Herbert, T. Kanade, and S Shafer. Vision and navigation for the carnegie-mellon navlab. *IEEE Trans. Pattern Analysis and Machine Intelligence*, 10(3):362–372, May 1988.

[15] S. Thrun. Learning metric-topological maps for indoor mobile robot navigation. *Artificial Intelligence*, 99(1):21–71, February 1998.
